# Processing of Visual and Auditory Space and Its Modification by Experience

**Josef P. Rauschecker**
Laboratory of Neurophysiology
National Institute of Mental Health
Poolesville, MD 20837

**Terrence J. Sejnowski**
Computational Neurobiology Lab
The Salk Institute
San Diego, CA 92138

Visual spatial information is projected from the retina to the brain in a highly topographic fashion, so that 2-D visual space is represented in a simple retinotopic map. Auditory spatial information, by contrast, has to be computed from binaural time and intensity differences as well as from monaural spectral cues produced by the head and ears. Evaluation of these cues in the central nervous system leads to the generation of neurons that are sensitive to the location of a sound source in space ("spatial tuning") and, in some animal species, to auditory space maps where spatial location is encoded as a 2-D map just like in the visual system. The brain structures thought to be involved in the multimodal integration of visual and auditory spatial integration are the superior colliculus in the midbrain and the inferior parietal lobe in the cerebral cortex.

It has been suggested for the owl that the visual system participates in setting up the auditory space map in the superior. Rearing owls with displacing prisms, for example, shifts the map by a fixed amount. These behavioral and neurobiological findings have been successfully incorporated into a connectionist model of the owl's sound localization system (Rosen, Rumelhart, and Knudsen, 1994). On the other hand, cats that are reared with both eyes sutured shut develop completely normal auditory spatial mechanisms: Precision of sound localization is even improved above normal (Rauschecker and Kniepert, 1994), and a higher number of auditory neurons with sharper spatial tuning is found in parietal cortex of such cats (Rauschecker and Korte, 1993). Non-visual sensory signals and/or motor feedback must be capable, therefore, to calibrate the auditory spatial mechanisms. Activity-dependent Hebbian learning and synaptic competition between inputs to the parietal region from different sensory modalities are sufficient to explain these results.

The question remains how visual and auditory information are kept in spatial register with each other when the animal moves its eyes or head. Experiments in awake behaving monkeys help to solve this problem. Neurons in the lateral intraparietal area of cortex (LIP) respond to visual and auditory stimuli which call for a movement to the same location in space. Neuronal responses in both modalities are modulated by eye position leading to "gain fields", in which the location of a target in head-centered coordinates is encoded via the response strength in a population of neurons (Andersen, Snyder, Li, and Stricanne, 1993).

The neurobiological data from owls, cats and monkeys were used to develop a neural network model of multisensory integration (Pouget and Sejnowski, 1993). A set of basis functions was introduced which replace the conventional allocentric representations and produce gain fields similar to monkey parietal cortex. An extension of the model also incorporates the plasticity of this system. Predictive Hebbian learning is used to bring the visual and auditory maps into register. In the network a Hebb rule is gated by a reinforcement term, which is the difference between actual reinforcement and how much reinforcement is expected by the system. It utilizes the activity of diffuse transmitter projection systems, such as noradrenaline (NA), acetylcholine (ACh), and dopamine (DA), which are known to play an important role for plasticity in the brain of higher mammals.

In summary, it appears extremely fruitful to bring together neuroscientists and neural network modelers, because both groups can profit from each other. Neurobiological data are the flesh for realistic network models, and models are helpful to formalize a biological hypothesis and guide the way for further testing.

Andersen RA, Snyder LH, Li C-S, Stricanne B (1993) Coordinate transformations in the representation of spatial information. *Curr Opinion Neurobiol* **3**:171-176.

Pouget A, Fisher SA, Sejnowski TJ (1993) Egocentric spatial representation in early vision. *J Cog Neurosci* **5**:150-161.

Rauschecker JP and Korte M (1993) Auditory compensation for early blindness in cat cerebral cortex. *J Neurosci* **13**:4538-4548.

Rauschecker JP and Kniepert U (1994) Enhanced precision of auditory localization behavior in visually deprived cats. *Eur J Neurosci* **6** (in press).

Rosen D, Rumelhart D, Knudsen E (1994) A connectionist model of the owl's sound localization system. In: *Advances in Neural Information Processing Systems* **6**, Cowan J, Tesauro G, Alspector J (eds), San Mateo, CA: Morgan Kaufmann (in press)
